# Manifold Regularization for SIR with Rate Root-n Convergence

**Wei Bian**
School of Computer Engineering
Nanyang Technological University
Singapore, 639798
weibian@pmail.ntu.edu.sg

**Dacheng Tao**
School of Computer Engineering
Nanyang Technological University
Singapore, 639798
dctao@ntu.edu.sg

## Abstract

In this paper, we study the manifold regularization for the Sliced Inverse Regression (SIR). The manifold regularization improves the standard SIR in two aspects: 1) it encodes the local geometry for SIR and 2) it enables SIR to deal with transductive and semi-supervised learning problems. We prove that the proposed graph Laplacian based regularization is convergent at rate root-n. The projection directions of the regularized SIR are optimized by using a conjugate gradient method on the Grassmann manifold. Experimental results support our theory.

## 1 Introduction

Sliced inverse regression (SIR) [7] was proposed for sufficient dimension reduction. In a regression setting, with the predictors X and the response Y, the sufficient dimension reduction (SDR) subspace $B$ is defined by the conditional independency $Y \perp X | B^{\mathrm{T}}X$. Under the assumption that the distribution of X is elliptic symmetric [7], it has been proved that the SDR subsapce $B$ is related to the inverse regression curve E(X|Y). It can be estimated at least partially by a generalized eigen-decomposition between the covariance matrix of the predictors Cov(X) and the covariance matrix of the inverse regression curve Cov(E(X|Y)). When Y is a continuous random variable, it is discretized by slicing its range into several slices so as to estimate E(X|Y) empirically. This procedure reflects the name of SIR.

For practical applications, the elliptic symmetric assumption on $P(X)$ in SIR cannot be fully satisfied, because many real datasets are embedded on manifolds [1]. Therefore, SIR cannot select an efficient subspace for predicting the response Y because the local geometry of the predictors X is ignored. Additionally, SIR only utilizes labeled (given response) data (predictors). Thus, it is valuable to extend SIR to deal with transductive and semi-supervised learning problems by considering unlabelled samples.

We solve the above two problems of SIR by using the manifold regularization [2], which has been developed to incorporate the local geometry in learning classification or regression functions. In this paper, we utilize it to preserve the local geometry of predictors in learning the SDR subspace $B$. In addition, it helps SIR to solve transductive/semi-supervised learning problems because the regularization encodes the marginal distribution of the unlabelled predictors.

Different regularizations for SIR have been well studied, e.g., the non-singular regularization [14], the ridge regularization [9], and the sparse regularization [8]. However, all existing regularizations do not encode the local geometry of the predictors. Although the localized sliced inverse regression [12] considers the local geometry, it is heuristic and does not follow up the regularization framework.

The rest of the paper is organized as following. Section 2 presents the manifold regularization for SIR. Section 3 proves the convergence of the new manifold regularization. We discuss the optimiza-

tion algorithm of the regularized SIR by using the conjugate gradient method on the Grassmann manifold in Section 4. Section 5 presents the experimental results on synthetic and real datasets. Section 6 concludes this paper.

## 2 Manifold Regularization for SIR

In the rest of the paper, we use terminologies in regression and deem classification as regression with the category response. Upper case letters $X \in R^p$ and $Y \in R$ are respectively the predictors and the response, and lower case letters $x$ and $y$ are corresponding realizations. Given a sample set, containing $n_l$ labeled samples $\{x_i, y_i\}_{i=1}^{n_l}$ and $n_u$ unlabeled samples $\{x_i\}_{i=n_l+1}^{n=n_l+n_u}$, we seek an optimal $k$-dimensional subspace spanned by $B = [\beta_1, ..., \beta_k]$ such that the response Y is predictable with the projected predictors $B^T X$. We also use matrix $X = [x_1, x_2, ..., x_n]$ to denote all predictors in the sample set.

### 2.1 Sliced Inverse Regression

Suppose the response Y is predictable with a sufficient $k$-dimensional projection of the original predictors X. We can consider the following regression model [7].

$$Y = f\left(\beta_1^T X, \beta_2^T X, ..., \beta_k^T X, \varepsilon\right) \tag{1}$$

where $\beta$'s are linear independent projection vectors and $\varepsilon$ is the independent noise. Given a set of samples $\{x_i, y_i\}_{i=1}^{n_l}$, SIR estimates the projection subspace $B = [\beta_1, ..., \beta_k]$ via following steps: discretize Y by slicing its range into $H$ slices; calculate the sample frequency $f_h$ of Y falling into the $h$-th slice and the sample estimation of the conditional mean $\bar{X}_h = E(X|Y = h)$; estimate the mean $\bar{X}$ and covariance matrix $\Sigma$ of predictors X; calculate the matrix $\Gamma = \sum_h f_h \left(\bar{X}_h - \bar{X}\right)\left(\bar{X}_h - \bar{X}\right)^T$; and $B$ is finally obtained by using the generalized eigen-decomposition $\Sigma\beta = \lambda\Gamma\beta$. It can be proved that the generalized eigen-decomposition is equivalent to the following optimization,

$$\max_B trace\left(\left(B^T\Sigma B\right)^{-1} B^T\Gamma B\right). \tag{2}$$

We refer to (2) as the objective function of SIR and thus we can impose with the manifold regularization on (2).

**Remark 2.1** Another way to get the objective (2) is based on the least square formulation for SIR proposed in [3],

$$\min_B L\left(B, C\right) = \sum_{h=1}^{H} f_h \left(\bar{X}_h - \bar{X} - \Sigma BC_h\right)^T \Sigma^{-1} \left(\bar{X}_h - \bar{X} - \Sigma BC_h\right) \tag{3}$$

where $C = [C_1, C_2, ..., C_h]$ are auxiliary variables. Eliminate $C_h$ by setting the partial derivative $\partial L/\partial C_h = 0$, and then (2) can be obtained directly. Additionally, (2) shows that SIR could have a similar objective as linear discriminant analysis, although they are obtained from different understandings of discriminative dimension reduction.

### 2.2 Manifold Regularization for SIR

Each dimension reduction projection $\beta$ can be deemed as a linear function or a mapping $g(x) = \beta^T x$. We expect to preserve the local geometry of the distribution of the predictors X while doing mapping $g(x)$. Suppose the predictors X are embedded on a manifold $M$, this can be achieved by penalizing the gradient $\nabla_M g$ along the manifold $M$. Because we are dealing with random variables with the distribution $P(X)$, the following formulation can be applied,

$$R = \int_{X \in M} \|\nabla_M g\|^2 dP(X). \tag{4}$$

The above formulation is different from the original manifold regularization [2]on the point that the function $g(x)$ is a dimension reduction mapping here while it is a classification or regression

function in [2]. Usually, both the manifold and the marginal distribution of X are unknown. It has been well studied in manifold learning, however, that the regularization (4) can be approximated by using the associated graph Laplacian of labeled and unlabeled $\{x_i\}_{i=1}^{n=n_l+n_u}$.

Construct an adjacent graph for $\{x_i\}_{i=1}^{n=n_1+n_u}$, where the pairwise edge weight $(W)_{ij} = \phi(\|x_i - x_j\|)$ is defined by the kernel function $\phi(\cdot)$, e.g., the heat kernel $\phi(d) = \exp(-d^2)$, and then the associated graph Laplacian is $L = D - W$, where $D$ is a diagonal matrix given by $D_{ii} = \sum_j W_{ij}$. Thus, the regularization in (4) can be approximated by $R = \mathbf{g}^T L \mathbf{g}$, where $\mathbf{g} = [\beta^T x_1, ..., \beta^T x_n]$. Furthermore, because there are $k$ independent projections $B = [\beta_1, ..., \beta_k]$, we take the summation of $k$ regularizations

$$R = \sum_{i=1}^{k} \mathbf{g}_i^T L \mathbf{g}_i = trace\left(G^T L G\right) \tag{5}$$

where $G = [\mathbf{g}_1, ..., \mathbf{g}_k]$.

In manifold learning, it is suggested to use the normalized graph Laplacian $D^{-1/2} L D^{-1/2}$ to replace $L$, or to use an equivalent constraint $G^T D G = I$, to get a better performance [1], and the solution obtained by the normalized graph Laplacian is consistent with weaker conditions than the unnormalized one [13]. In the proposed regularized SIR, we normalize the regularization (5) as $R = trace\left(\left(G^T D G\right)^{-1} G^T L G\right)$, which is equivalent to the constraint $G^T D G = I$. This normalization makes $R$ invariant to scalar and rotation transformations of the projections $B = [\beta_1, ..., \beta_k]$, which is preferred for dimension reduction problems. By adding the regularization $R = trace\left(\left(G^T D G\right)^{-1} G^T L G\right)$ to SIR (2), and substituting $G = X^T B$, we get the regularized SIR

$$\max_{B} SIR_r(B) = trace\left(\left(B^T \Sigma B\right)^{-1} B^T \Gamma B\right) - \eta\, trace\left(\left(B^T S B\right)^{-1} B^T Q B\right) \tag{6}$$

where $Q = 1/n\,(n-1)\,XLX^T$, $S = 1/n\,(n-1)\,XDX^T$, and $\eta$ is the positive weighting factor.

## 3 Convergence of the Regularization

Different from the existing regularizations [8,9,14] for SIR, which are constructed as deterministic terms, the manifold regularization in (6) is a random term that involves two data dependent variables (matrices) $Q$ and $S$. Therefore, it is necessary to discuss the convergence property of the proposed manifold regularization.

It has been well proved that both $\Sigma$ and $\Gamma$ converge at rate root-n [7,11,15]. Therefore, the convergence rate of the objective (6) depends on whether the regularization term converges at rate root-n. Below, we prove that both the sample based estimations $Q = 1/n\,(n-1)\,XLX^T$ and $S = 1/n\,(n-1)\,XDX^T$ converge to deterministic matrices at rate root-n. Note that the convergence of a special case where the graph Laplacian is built by the kernel function $\phi(d) = 1\,(d < \varepsilon)$ was proved in [6]. Our proof scheme, however, is quite other than that used in [6]. Additionally, we target a general choice of kernel $\phi(\cdot)$ and also prove the root-n convergence rate which has not been obtained before.

Although samples $\{x_i\}_{i=1}^{n=n_l+n_u}$ are independent, the dependency of $L$ and $D$ on samples makes $Q$ and $S$ cannot be expanded as a summation of independent items. Therefore, it is difficult to apply the law of large numbers and the central limit theorem to prove the convergence and obtain the corresponding convergence rate. However, we can prove them by constructing the converged limitation and show that the variance of the sample based estimation with respect to the constructed limitation decades at rate root-n. Throughout the results obtained in this Section, we assume the following conditions hold.

**Conditions 3.1** *For kernel function $\phi(d)$, it satisfies $\phi(0) = 1$ and $|\phi(d)| \leqslant 1$. For the distribution of predictors $P(\mathrm{X})$, the fourth order moment exists, i.e., $E\left(\left\|\left(vec(xx^T)\right)\left(vec(xx^T)\right)^T\right\|\right) < \infty$, where $vec()$ vectorizes a matrix into a column vector.*

We start by splitting $Q$ into two parts $T_1$ and $T_2$,

$$Q = \frac{1}{n(n-1)} XLX^T = \frac{1}{n(n-1)} \sum_{i=1}^{n} (D_{ii} - W_{ii}) x_i x_i^T - \frac{1}{n(n-1)} \sum_{i \neq j}^{n} W_{ij} x_i x_j^T = T_1 - T_2. \quad (7)$$

Substituting the function $\phi(\cdot)$ into (7), we have

$$\begin{cases} T_1 = \frac{1}{n(n-1)} \sum_{i=1}^{n} \left( \sum_{j=1}^{n} \phi(\|x_i - x_j\|) - \phi(0) \right) x_i x_i^T = \frac{1}{n} \sum_{i=1}^{n} \left( \frac{1}{n-1} \sum_{j \neq i}^{n} \phi(\|x_i - x_j\|) \right) x_i x_i^T \\ T_2 = \frac{1}{n(n-1)} \sum_{i \neq j}^{n} W_{ij} x_i x_j^T = \frac{1}{n} \sum_{i}^{n} x_i \left( \frac{1}{n-1} \sum_{j \neq i}^{n} \phi(\|x_i - x_j\|) x_j^T \right). \end{cases} \quad (8)$$

Under the condition 3.1, the next two lemmas show the convergence of $T_1$ and $T_2$, respectively.

**Lemma 3.1** *Let the conditional expectation $\varphi(x) = E(\phi(\|z - x\|)\,|x)$, where $z$ and $x$ are independent and both are sampled from $P(\mathrm{X})$. The $E(\varphi(x) xx^T)$ exists, and $T_1$ in (8) converges almost surely at rate $n^{-1/2}$, i.e.,*

$$T_1 \overset{a.s}{=} E(\varphi(x) xx^T) + O\left(n^{-1/2}\right). \quad (9)$$

**Lemma 3.2** *Let the conditional expectation $\eta(x) = E(\phi(\|z - x\|) z\,|x)$, where $z$ and $x$ are independent and both are sampled from $P(\mathrm{X})$. The $E\left(x\eta(x)^T\right)$ exists, and $T_2$ in (8) converges almost surely at rate $n^{-1/2}$, i.e.,*

$$T_2 \overset{a.s}{=} E\left(x\eta(x)^T\right) + O\left(n^{-1/2}\right). \quad (10)$$

The proofs of above two lemmas are given in Section 6. Based on Lemmas 1 and 2, we have the following two theorems for the convergence of $Q$ and $S$.

**Theorem 3.1** *Given the Conditions 3.1, the sample based estimation $Q$ converges almost surely to a deterministic matrix $E(Q) = E(\varphi(x) xx^T) - E\left(x\eta(x)^T\right)$ at rate $n^{-1/2}$, i.e., $Q \overset{a.s}{=} E(Q) + O\left(n^{-1/2}\right)$.*

*Proof.* Because $Q = T_1 - T_2$, the theorem is an immediate result from Lemmas 3.1 and 3.2.

**Theorem 3.2** *Given the Conditions 3.1, the sample based estimation $S$ converges almost surely to a deterministic matrix $E(\varphi(x) xx^T)$ at rate $n^{-1/2}$, i.e., $S \overset{a.s}{=} E(\varphi(x) xx^T) + O\left(n^{-1/2}\right)$.*

*Proof.* $D_{ii} = \sum_j W_{ij} = \sum_{j=1}^{n} \phi(\|x_i - x_j\|)$, so $S = \frac{1}{n(n-1)} \sum_{i=1}^{n} D_{ii} x_i x_i^T = $
$\frac{1}{n(n-1)} \sum_{i=1}^{n} \left( \sum_{j=1}^{n} \phi(\|x_i - x_j\|) \right) x_i x_i^T = \frac{1}{n(n-1)} \sum_{i=1}^{n} \left( \sum_{j \neq i} \phi(\|x_i - x_j\|) + \phi(0) \right) x_i x_i^T = T_1 + $
$\frac{1}{n(n-1)} \sum_{i=1}^{n} x_i x_i^T$. Because $\frac{1}{(n-1)} \sum_{i=1}^{n} x_i x_i^T$ is an unbiased estimation of $\mathrm{Cov}(X)$, we have
$\frac{1}{n(n-1)} \sum_{i=1}^{n} x_i x_i^T \overset{a.s.}{=} O\left(n^{-1}\right)$. Therefore, according to Lemma 3.1, we have $S = T_1 + $
$O\left(n^{-1}\right) \overset{a.s.}{=} E(\varphi(x) xx^T) + O\left(n^{-1/2}\right)$. Note that here $E(S) \neq E(\varphi(x) xx^T)$, but equality can be asymptotically achieved when $n \to \infty$.

## 4  Optimization on the Grassmann Manifold

The optimization of the regularized SIR (6) is much more difficult than that of the standard SIR (2), which can be solved by the generalized eigen-decomposition. In this section, we present a conjugate

gradient method on the Grassmann manifold to solve (6), based on the fact it is invariant to scalar and rotation transformations of the projection $B$. By exploiting the geometry of the Grassmann manifold, the conjugate gradient algorithm converges faster than the gradient scheme in the Euclidean space.

Given a constrained optimization problem $\min F(A)$ subject to $A \in R^{p \times k}$ and $A^T A = I$, if the problem further satisfies $F(A) = F(AO)$ for an arbitrary orthonormal matrix $O$, then it is called an optimization problem defined on the Grassmann manifold $\mathcal{G}_{pk}$. By the following theorem, we can transform (6) into its equivalent form (11) which is defined on the Grassmann manifold.

**Theorem 4.1** *Suppose that $\Sigma$ is nonsingular and given the eigen-decomposition $\Sigma^{-1/2} S \Sigma^{-1/2} = U \tilde{\Lambda} U^T$, problem (6) is equivalent to*

$$\min_{A^T A = I} F(A) = -trace\left(A^T \tilde{\Gamma} A\right) + \eta trace\left(\left(A^T \tilde{\Lambda} A\right)^{-1} A^T \tilde{Q} A\right) \tag{11}$$

*where $\tilde{\Gamma} = U^T \Sigma^{-1/2} \Gamma \Sigma^{-1/2} U$ and $\tilde{Q} = U^T \Sigma^{-1/2} Q \Sigma^{-1/2} U$. Given the optimal solution $A$ of (11), the optimal solution of (6) is given by $B = \Sigma^{-1/2} U A$.*

*Proof.* Substituting $B = \Sigma^{-1/2} U A$ into (6), we have $SIR_r(A) = trace\left(\left(A^T A\right)^{-1} A^T \tilde{\Gamma} A\right) - \eta trace\left(\left(A^T \tilde{\Lambda} A\right)^{-1} A^T \tilde{Q} A\right)$. Given a nonsingular $\Sigma$, $B = \Sigma^{-1/2} U A$ is an invertible variable transform. Thus, we know that if $A$ maximizes $SIR_r(A)$ then $B$ maximizes $SIR_r(B)$. Because $SIR_r(A)$ is invariant to scalar and rotation transformations, a constraint $A^T A = I$ can be added to (6). We then get (11). This completes the proof.

To implement the conjugate gradient method on the Grassmann manifold, the gradient of $F(A)$ in (11) is required. According to [4], the gradient $G_A$ of $F(A)$ on the manifold is defined by $G_A = \Pi_A F_A$ where $F_A$ is the gradient of $F(A)$ in the Euclidian space and $\Pi_A = I - AA^T$ is the projection onto the tangent space at $A$ of the manifold. In case of $F(A)$ in (11), it is given by,

$$G_A = \left(I - AA^T\right) \tilde{\Gamma} A - \eta \left(I - \tilde{\Lambda} A \left(A^T \tilde{\Lambda} A\right)^{-1} A^T\right) \hat{Q} A \left(A^T \tilde{\Lambda} A\right)^{-1}. \tag{12}$$

Next, we present the conjugate gradient method on the Grassmann manifold [4] to solve (11). The algorithm is given by the following three steps:

- **1-D searching along the geodesic**: given the current position $A_k$, the gradient $G_k$ and the searching direction $H_k$, the 1-D searching along the geodesic is given by

$$\min_t F(A(t)) \text{ s.t. } A(t) = F\left(A_k V \cos(\Sigma t) V^T + U \sin(\Sigma t) V^T\right) \tag{13}$$

  where $U\Sigma V^T$ is the compact SVD of $H_k$. Record the minimum solution $t_k = t_{\min}$, and $A_{k+1} = A(t_k)$ as the starting position for next searching.

- **Transporting gradient and search direction**: parallel transport $G_k$ and $H_k$ from $A_k$ to $A_{k+1}$ by using

$$\tau G_k = G_k - \left(A_k V \sin \Sigma t_k + U\left(I - \cos \Sigma t_k\right)\right) U^T G_k \tag{14}$$

$$\tau H_k = \left(-A_k V \sin \Sigma t_k + U \cos \Sigma t_k\right) \Sigma V^T \tag{15}$$

- **Calculating the conjugate direction**: given the gradient $G_{k+1}$ at $A_{k+1}$, the conjugate searching direction is

$$H_{k+1} = -G_{k+1} + trace\left(\left(G_{k+1} - \tau G_k\right)^T G_{k+1}\right) / trace\left(G_k^T G_k\right) \tau H_k. \tag{16}$$

Initialize $A_0$ by a random guess (subject to $A_0^T A_0 = I$) and let $H_0 = -G_0$, and then repeat the above three steps iteratively to minimize $F(A)$ until convergence, i.e., $|F(A_{k+1}) - F(A_k)| < \varepsilon_0$. Note that, the same as the conjugate gradient method in the Euclidian space, the searching direction $H_k$ has to be resetting as $H_k = -G_k$ with a period of $p(n-p)$, i.e., the dimension of the searching space.

# 5 Experiments

In this section, we evaluate the proposed regularized SIR on two real datasets. We show the results of the standard SIR and the localized SIR on the same experiments for reference.

## 5.1 USPS Test

The USPS dataset contains 9,298 handwriting characters of digits 0 to 9. The entire USPS database is divided into two parts, a training set is with 7,291 samples and a test set is with 2,007 samples [5]. In our experiment, dimension reduction is first implemented and then the nearest neighbor rule is used for classification. By using the 1/3 of the data in training set as labeled data and the rest 2/3 as unlabeled data, we conduct supervised and semisupervised dimension reduction by the following five methods: supervised training of standard SIR, the manifold regularized SIR, and the localized SIR, and semi-supervised training of the manifold regularized SIR and the localized SIR. Performances are evaluated on the independent testing set. Table 1 summarizes the experimental results. It shows that both the regularized SIR and the localized SIR [12] can achieve superior performance to the standard SIR, and the manifold regularized SIR performs better than the localized SIR in both the supervised and the semi-supervised training. Experimental results reflect that the manifold regularized SIR is effective on exploiting the local geometry of a dataset.

Table 1: Experimental results on the USPS dataset: SIR; the manifold regularized SIR (RSIR); the localized SIR (LSIR); semi-supervised training of the manifold regularized SIR (sRSIR); semi-supervised training of the localized SIR (sLSIR).

| Dimensionality | 7 | 9 | 11 | 13 | 15 | 17 | 19 | 21 |
|---|---|---|---|---|---|---|---|---|
| SIR | 0.8635 | 0.8794 | — | — | — | — | — | — |
| RSIR | 0.8575 | 0.8809 | 0.8859 | 0.8889 | 0.9028 | 0.9108 | 0.9148 | 0.9193 |
| sRSIR | 0.8685 | 0.8864 | 0.8934 | 0.8909 | 0.9053 | 0.9128 | 0.9208 | 0.9193 |
| LSIR | 0.8301 | 0.8421 | 0.8535 | 0.8724 | 0.8789 | 0.8949 | 0.8989 | 0.9003 |
| sLSIR | 0.8526 | 0.8675 | 0.8795 | 0.8826 | 0.8914 | 0.8954 | 0.9038 | 0.9063 |

## 5.2 Transductive Visualization

In Coil-20 database [10], each object has 72 images taken from different view angles. All images are cropped into 128×128 pixel arrays with 256 gray levels. We then reduce the size to 32×32, and used the first 10 objects for 2-D visualization, with randomly labeled 6 out of 72 images. Figure 1 shows the visualization results obtained by SIR, the proposed regularized SIR and the localized SIR [12]. The figure shows that by exploiting the unlabeled data via the manifold regularization for dimension reduction, the performance for data visualization can be significantly improved. The localized SIR performs better than SIR, but not as good as the regularized SIR.

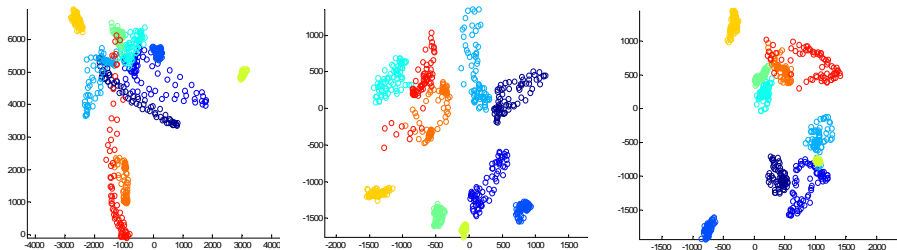

Figure 1: Visualization of the first 10 objects in Coil-20 database: from left to right, by the standard SIR, the manifold regularized SIR, and the localized SIR.

# 6 Proofs of Lemmas

**Proof of Lemma 3.1** Because the kernel function $\phi(\cdot)$ is bounded by $|\phi(d)| \leqslant 1$, we have $|\varphi(x)| = |E(\phi(\|z - x\|)|x)| \leqslant 1$, which implies that $E(\varphi(x)xx^T)$ exists. Then, to prove $T_1 \overset{a.s}{=} E(\varphi(x)xx^T) + O(n^{-1/2})$, it is sufficient to show that $E(T_1) = E(\varphi(x)xx^T)$ and $\mathrm{Cov}(vec(T_1)) = E((vec(T_1))(vec(T_1))^T) - (vec(E(T_1)))(vec(E(T_1)))^T = O(n^{-1})$. First, because $x_i$ and $x_j$ are independent when $i \neq j$, it follows,

$$
\begin{aligned}
E(T_1) &= E\left(\frac{1}{n}\sum_{i=1}^{n}\left(\frac{1}{n-1}\sum_{j\neq i,j=1}^{n}\phi(\|x_i - x_j\|)\right)x_i x_i^T\right) \\
&= \frac{1}{n}\sum_{i=1}^{n}E\left(x_i x_i^T\left(\frac{1}{n-1}\sum_{j\neq i,j=1}^{n}E(\phi(\|x_i - x_j\|)|x_i)\right)\right) \\
&= \frac{1}{n}\sum_{i=1}^{n}E\left(x_i x_i^T\varphi(x_i)\right) = E\left(\varphi(x)xx^T\right).
\end{aligned}
\tag{17}
$$

Next, we show $E\left((vec(T_1))(vec(T_1))^T\right)$ is a summation of two terms, of which one is $(vec(E(T_1)))(vec(E(T_1)))^T$ and the other is $O(n^{-1})$.

$$
\begin{aligned}
&E\left((vec(T_1))(vec(T_1))^T\right) \\
&= \frac{1}{n^2(n-1)^2}E\left(vec\left(\sum_{i\neq j}^{n}\phi(\|x_i - x_j\|)x_i x_i^T\right)\left(vec\left(\sum_{i\neq j}^{n}\phi(\|x_i - x_j\|)x_i x_i^T\right)\right)^T\right) \\
&= \frac{1}{n^2(n-1)^2}\sum_{i\neq j}^{n}\sum_{i'\neq j'}^{n}E\left(vec\left(\phi(\|x_i - x_j\|)x_i x_i^T\right)\left(vec\left(\phi(\|x_{i'} - x_{j'}\|)x_{i'}x_{i'}^T\right)\right)^T\right) \\
&= \frac{1}{n^2(n-1)^2}\sum_{i,j,i',j'\,distinct}E(\Phi_{i,j,i',j'}) + \frac{1}{n^2(n-1)^2}\sum_{else}E(\Phi_{i,j,i',j'}),
\end{aligned}
\tag{18}
$$

where $\Phi_{i,j,i',j'} = vec\left(\phi(\|x_i - x_j\|)x_i x_i^T\right)\left(vec\left(\phi(\|x_{i'} - x_{j'}\|)x_{i'}x_{i'}^T\right)\right)^T$.

When $i, j, i', j'$ are distinct, $x_i, x_j, x_{i'}$, and $x_{j'}$ are independent, we have

$$
\begin{aligned}
E(\Phi_{i,j,i',j'}) &= E\left((vec(\phi(\|x_i - x_j\|)x_i x_i^T))(vec(\phi(\|x_{i'} - x_{j'}\|)x_{i'}x_{i'}^T))^T\right) \\
&= (vec(E(\varphi(x)xx^T)))(vec(E(\varphi(x)xx^T)))^T \\
&= (vec(E(T_1)))(vec(E(T_1)))^T.
\end{aligned}
\tag{19}
$$

Therefore, the first term in $E\left(vec(T_1)(vec(T_1))^T\right)$ is

$$
\begin{aligned}
\frac{1}{n^2(n-1)^2}\sum_{\substack{i,j,i',j'\\distinct}}E(\Phi_{i,j,i',j'}) &= \frac{n(n-1)(n-2)(n-3)}{n^2(n-1)^2}(vec(E(T_1)))(vec(E(T_1)))^T \\
&= (vec(E(T_1)))(vec(E(T_1)))^T + O(n^{-1}).
\end{aligned}
\tag{20}
$$

For the second term in $E\left((vec(T_1))(vec(T_1))^T\right)$, $E(\Phi_{i,j,i',j'})$ is bounded by a constant (matrix) $M$ under the Conditions 3.1, and thus we have

$$
\left|\frac{1}{n^2(n-1)^2}\sum_{else}E(\Phi_{i,j,i',j'})\right| \leqslant \frac{1}{n^2(n-1)^2}\sum_{else}M = \frac{n(n-1)(4n-6)}{n^2(n-1)^2}M = O(n^{-1}). \tag{21}
$$

Combining the above two results, we have

$$\text{Cov}\left(vec\left(T_1\right)\right) = E(vec\left(T_1\right))\left(vec\left(T_1\right)\right)^T - \left(vec\left(E\left(T_1\right)\right)\right)\left(vec\left(E\left(T_1\right)\right)\right)^T = O\left(n^{-1}\right) \quad (22)$$

**Proof of Lemma 3.2** Similar to the proof of Lemma 3.1, $E\left(x\eta\left(x\right)^T\right)$ exists. Then, it is sufficient to show that $E\left(T_2\right) = E\left(x\eta\left(x\right)^T\right)$ and $\text{Cov}\left(vec\left(T_2\right)\right) = O\left(n^{-1}\right)$. First, we have

$$
\begin{aligned}
E\left(T_2\right) &= E\left(\frac{1}{n}\sum_{i=1}^n x_i \left(\frac{1}{n-1}\sum_{j\neq i,j=1}^n \phi\left(\|x_i - x_j\|\right)x_j^T\right)\right) \\
&= \frac{1}{n}\sum_{i=1}^n E\left(x_i\left(\frac{1}{n-1}\sum_{j\neq i,j=1}^n E\left(\phi\left(\|x_i - x_j\|\right)x_j^T \,|x_i\right)\right)\right) \\
&= \frac{1}{n}\sum_{i=1}^n E\left(x_i\eta\left(x_i\right)\right) = E\left(x\eta\left(x\right)\right).
\end{aligned} \quad (23)
$$

Next, we split $E\left(\left(vec\left(T_2\right)\right)\left(vec\left(T_2\right)\right)^T\right)$ into two terms

$$
\begin{aligned}
&E\left(\left(vec\left(T_2\right)\right)\left(vec\left(T_2\right)\right)^T\right) \\
&= \frac{1}{n^2\left(n-1\right)^2}E\left(vec\left(\sum_{i\neq j}\phi\left(\|x_i - x_j\|\right)x_i x_j^T\right)\left(vec\left(\sum_{i\neq j}\phi\left(\|x_i - x_j\|\right)x_i x_j^T\right)\right)^T\right) \\
&= \frac{1}{n^2\left(n-1\right)^2}\left(\sum_{i\neq j}^n\sum_{i'\neq j'}^n E\left(\left(vec\left(\phi\left(\|x_i - x_j\|\right)x_i x_j^T\right)\right)\left(vec\left(\phi\left(\|x_{i'} - x_{j'}\|\right)x_{i'} x_{j'}^T\right)\right)^T\right)\right) \\
&= \frac{1}{n^2\left(n-1\right)^2}\sum_{i,j,i',j'\,distinct}E\left(\Psi_{i,j,i',j'}\right) + \frac{1}{n^2\left(n-1\right)^2}\sum_{else}E\left(\Psi_{i,j,i',j'}\right)
\end{aligned} \quad (24)
$$

where $\Psi_{i,j,i',j'} = vec\left(\phi\left(\|x_i - x_j\|\right)x_i x_j^T\right)\left(vec\left(\phi\left(\|x_{i'} - x_{j'}\|\right)x_{i'} x_{j'}^T\right)\right)^T$.

Following the same method used in the proof of Lemma 3.1, we have

$$\frac{1}{n^2\left(n-1\right)^2}\sum_{else}E\left(\Psi_{i,j,i',j'}\right) = \left(vec\left(E\left(T_2\right)\right)\right)\left(vec\left(E\left(T_2\right)\right)\right)^T + O\left(n^{-1}\right) \quad (25)$$

and

$$\left|\frac{1}{n^2\left(n-1\right)^2}\sum_{else}E\left(\Psi_{i,j,i',j'}\right)\right| \leqslant O\left(n^{-1}\right). \quad (26)$$

Therefore, we have $\text{Cov}\left(vec\left(T_2\right)\right) = O\left(n^{-1}\right)$.

# 7 Conclusion

We have studied the manifold regularization for Sliced Inverse Regression (SIR). The regularized SIR extended the original SIR in many ways, i.e., it utilizes the local geometry that is ignored originally and enables SIR to deal with the tranductive/semisupervised learning problems. We also discussed the statistical properties of the proposed regularization, that under mild conditions, the manifold regularization converges at rate root-n. To solve the regularized SIR problem, we present a conjugate gradient method conducted on the Grassmann manifold. Experiments on real datasets validate the effectiveness of the regularized SIR.

**Acknowledgments**

This project was supported by the Nanyang Technological University Nanyang SUG Grant (under project number M58020010).

**References**

[1] Belkin, M. & Niyogi, P. (2003) Laplacian eigenmaps for dimensionality reduction and data representation. *Neural Computation*, 15(6): 1373-1396.

[2] Belkin, M., Niyogi, P. & Sindhwani, V. (2006) Manifold regularization: Ageometric framework for learning from labeled and unlabeled examples. *Journal of Machine Learning Research*, 1: 1-48.

[3] Cook, R.D.(2004) Testing predictor contributions in sufficient dimension reduction. *Annals of Statistics*, 32: 1061-1092.

[4] Edelman, A., Arias, T.A., & Smith, S.T. (1998) The geometry of algorithms with orthogonality constraints. *SIAM J. Matrix Anal. Appl.*, 20(2):303-353.

[5] Hastie, T., Buja, A., & Tibshirani, R. (1995) Penalized discriminant analysis. *Annals of Statistics*, 2: 73-102.

[6] He, X., Deng, C., & Min, W. (2005) Statistical and computational analysis of locality preserving projection. In *22th International Conference on Machine Learning (ICML)*.

[7] Li, K. (1991) Sliced inverse regression for dimension reduction (with discussion). *J. Amer. Statist. Assoc.*, 86:316-342.

[8] Li, L. (2007). Sparse sufficient dimension reduction. *Biometrika* 94(3): 603-613.

[9] Li, L., & YIN, X. (2008). Sliced inverse regression with regularizations. *Biometrics* 64: 124-131.

[10] Nene, S.A., Nayar, S.K., & Murase, H. (1996) Columbia object image library: COIL-20. Technical Report No. CUCS-006-96, Dept. of Computer Science, Columbia University.

[11] Saracco, J. (1997). An asymptotic theory for sliced inverse regression. *Comm. Statist. Theory Methods* 26: 2141-2171.

[12] Wu, Q., Mukherjee, S., & Liang, F. (2008) Localized sliced inverse regression. *Advances in neural information processing systems 20*, Cambridge, MA: MIT Press.

[13] von Luxburg, U., Bousquet, O., & Belkin, M. (2005) Limits of spectral clustering. In L. K. Saul, Y. Weiss and L. Bottou (Eds.), *Advances in neural information processing systems 17*, Cambridge, MA: MIT Press.

[14] Zhong, W., Zeng, P., Ma, P., Liu, J. S., & Zhu, Y. (2005) RSIR: Regularized sliced inverse regression for motif discovery. *Bioinformatics* 21: 4169-4175.

[15] Zhu, L.X., & NG, K.W. (1995) Asymptotics of sliced inverse regression. *Statistica Sinica* 5: 727-736.

